# A Dirty Model for Multi-task Learning

**Ali Jalali**
University of Texas at Austin
alij@mail.utexas.edu

**Pradeep Ravikumar**
University of Texas at Asutin
pradeepr@cs.utexas.edu

**Sujay Sanghavi**
University of Texas at Austin
sanghavi@mail.utexas.edu

**Chao Ruan**
University of Texas at Austin
ruan@cs.utexas.edu

## Abstract

We consider multi-task learning in the setting of multiple linear regression, and where some relevant features could be shared across the tasks. Recent research has studied the use of $\ell_1/\ell_q$ norm block-regularizations with $q > 1$ for such block-sparse structured problems, establishing strong guarantees on recovery even under high-dimensional scaling where the number of features scale with the number of observations. However, these papers also caution that the performance of such block-regularized methods are very dependent on the *extent* to which the features are shared across tasks. Indeed they show [8] that if the extent of overlap is less than a threshold, or even if parameter *values* in the shared features are highly uneven, then block $\ell_1/\ell_q$ regularization could actually perform *worse* than simple separate elementwise $\ell_1$ regularization. Since these caveats depend on the unknown true parameters, we might not know when and which method to apply. Even otherwise, we are far away from a realistic multi-task setting: not only do the set of relevant features have to be exactly the same across tasks, but their values have to as well.

Here, we ask the question: can we leverage parameter overlap when it exists, but not pay a penalty when it does not ? Indeed, this falls under a more general question of whether we can model such *dirty data* which may not fall into a single neat structural bracket (all block-sparse, or all low-rank and so on). With the explosion of such dirty high-dimensional data in modern settings, it is vital to develop tools – *dirty models* – to perform biased statistical estimation tailored to such data. Here, we take a first step, focusing on developing a dirty model for the multiple regression problem. Our method uses a very simple idea: we estimate a *superposition* of two sets of parameters and regularize them differently. We show both theoretically and empirically, our method strictly and noticeably outperforms both $\ell_1$ or $\ell_1/\ell_q$ methods, under high-dimensional scaling and over the entire range of possible overlaps (except at boundary cases, where we match the best method).

## 1 Introduction: Motivation and Setup

*High-dimensional scaling.* In fields across science and engineering, we are increasingly faced with problems where the number of variables or features $p$ is larger than the number of observations $n$. Under such high-dimensional scaling, for any hope of statistically consistent estimation, it becomes vital to leverage any potential structure in the problem such as sparsity (e.g. in compressed sensing [3] and LASSO [14]), low-rank structure [13, 9], or sparse graphical model structure [12]. It is in such high-dimensional contexts in particular that multi-task learning [4] could be most useful. Here,

multiple tasks share some common structure such as sparsity, and estimating these tasks jointly by leveraging this common structure could be more statistically efficient.

*Block-sparse Multiple Regression.* A common multiple task learning setting, and which is the focus of this paper, is that of multiple regression, where we have $r > 1$ response variables, and a common set of $p$ features or covariates. The $r$ tasks could share certain aspects of their underlying distributions, such as common variance, but the setting we focus on in this paper is where the response variables have *simultaneously sparse* structure: the index set of relevant features for each task is sparse; and there is a large overlap of these relevant features across the different regression problems. Such "simultaneous sparsity" arises in a variety of contexts [15]; indeed, most applications of sparse signal recovery in contexts ranging from graphical model learning, kernel learning, and function estimation have natural extensions to the simultaneous-sparse setting [12, 2, 11].

It is useful to represent the multiple regression parameters via a matrix, where each column corresponds to a task, and each row to a feature. Having simultaneous sparse structure then corresponds to the matrix being largely "block-sparse" – where each row is either all zero or mostly non-zero, and the number of non-zero rows is small. A lot of recent research in this setting has focused on $\ell_1/\ell_q$ norm regularizations, for $q > 1$, that encourage the parameter matrix to have such block-sparse structure. Particular examples include results using the $\ell_1/\ell_\infty$ norm [16, 5, 8], and the $\ell_1/\ell_2$ norm [7, 10].

*Dirty Models.* Block-regularization is "heavy-handed" in two ways. By strictly encouraging shared-sparsity, it assumes that all relevant features are shared, and hence suffers under settings, arguably more realistic, where each task depends on features specific to itself in addition to the ones that are common. The second concern with such block-sparse regularizers is that the $\ell_1/\ell_q$ norms can be shown to encourage the entries in the non-sparse rows taking nearly identical *values*. Thus we are far away from the original goal of multitask learning: not only do the set of relevant features have to be exactly the same, but their values have to as well. Indeed recent research into such regularized methods [8, 10] caution against the use of block-regularization in regimes where the supports and values of the parameters for each task can vary widely. Since the true parameter values are unknown, that would be a worrisome caveat.

We thus ask the question: can we learn multiple regression models by leveraging whatever overlap of features there exist, and without requiring the parameter values to be near identical? Indeed this is an instance of a more general question on whether we can estimate statistical models where the data may not fall cleanly into any one structural bracket (sparse, block-sparse and so on). With the explosion of *dirty* high-dimensional data in modern settings, it is vital to investigate estimation of corresponding *dirty models*, which might require new approaches to biased high-dimensional estimation. In this paper we take a first step, focusing on such dirty models for a specific problem: simultaneously sparse multiple regression.

Our approach uses a simple idea: while any one structure might not capture the data, a superposition of structural classes might. Our method thus searches for a parameter matrix that can be *decomposed* into a row-sparse matrix (corresponding to the overlapping or shared features) and an elementwise sparse matrix (corresponding to the non-shared features). As we show both theoretically and empirically, with this simple fix we are able to leverage any extent of shared features, while allowing disparities in support and values of the parameters, so that we are *always* better than both the Lasso or block-sparse regularizers (at times remarkably so).

The rest of the paper is organized as follows: In Sec 2. basic definitions and setup of the problem are presented. Main results of the paper is discussed in sec 3. Experimental results and simulations are demonstrated in Sec 4.

**Notation:** For any matrix $M$, we denote its $j^{th}$ row as $M_j$, and its $k$-th column as $M^{(k)}$. The set of all non-zero rows (i.e. all rows with at least one non-zero element) is denoted by $\text{RowSupp}(M)$ and its support by $\text{Supp}(M)$. Also, for any matrix $M$, let $\|M\|_{1,1} := \sum_{j,k} |M_j^{(k)}|$, i.e. the sums of absolute values of the elements, and $\|M\|_{1,\infty} := \sum_j \|M_j\|_\infty$ where, $\|M_j\|_\infty := \max_k |M_j^{(k)}|$.

## 2   Problem Set-up and Our Method

*Multiple regression.* We consider the following standard multiple linear regression model:

$$y^{(k)} = X^{(k)}\bar{\theta}^{(k)} + w^{(k)}, \quad k = 1, \ldots, r,$$

where $y^{(k)} \in \mathbb{R}^n$ is the response for the $k$-th task, regressed on the design matrix $X^{(k)} \in \mathbb{R}^{n \times p}$ (possibly different across tasks), while $w^{(k)} \in \mathbb{R}^n$ is the noise vector. We assume each $w^{(k)}$ is drawn independently from $\mathcal{N}(0, \sigma^2)$. The total number of tasks or target variables is $r$, the number of features is $p$, while the number of samples we have for each task is $n$. For notational convenience, we collate these quantities into matrices $Y \in \mathbb{R}^{n \times r}$ for the responses, $\bar{\Theta} \in \mathbb{R}^{p \times r}$ for the regression parameters and $W \in \mathbb{R}^{n \times r}$ for the noise.

*Dirty Model.* In this paper we are interested in estimating the true parameter $\bar{\Theta}$ from data by leveraging any (unknown) extent of simultaneous-sparsity. In particular, certain rows of $\bar{\Theta}$ would have many non-zero entries, corresponding to features shared by several tasks ("shared" rows), while certain rows would be elementwise sparse, corresponding to those features which are relevant for some tasks but not all ("non-shared rows"), while certain rows would have all zero entries, corresponding to those features that are not relevant to any task. We are interested in estimators $\widehat{\Theta}$ that automatically adapt to different levels of sharedness, and yet enjoy the following guarantees:

**Support recovery:** We say an estimator $\widehat{\Theta}$ successfully recovers the true signed support if $\text{sign}(\text{Supp}(\widehat{\Theta})) = \text{sign}(\text{Supp}(\bar{\Theta}))$. We are interested in deriving sufficient conditions under which the estimator succeeds. We note that this is stronger than merely recovering the row-support of $\bar{\Theta}$, which is union of its supports for the different tasks. In particular, denoting $\mathcal{U}_k$ for the support of the $k$-th column of $\bar{\Theta}$, and $\mathcal{U} = \bigcup_k \mathcal{U}_k$.

**Error bounds:** We are also interested in providing bounds on the elementwise $\ell_\infty$ norm error of the estimator $\widehat{\Theta}$,

$$\|\widehat{\Theta} - \bar{\Theta}\|_\infty = \max_{j=1,\ldots,p} \max_{k=1,\ldots,r} \left| \widehat{\Theta}_j^{(k)} - \bar{\Theta}_j^{(k)} \right|.$$

### 2.1   Our Method

Our method explicitly models the *dirty block-sparse* structure. We estimate a sum of two parameter matrices $B$ and $S$ with different regularizations for each: encouraging block-structured row-sparsity in $B$ and elementwise sparsity in $S$. The corresponding "clean" models would either just use block-sparse regularizations [8, 10] or just elementwise sparsity regularizations [14, 18], so that either method would perform better in certain suited regimes. Interestingly, as we will see in the main results, by explicitly allowing to have both block-sparse and elementwise sparse component, we are able to *outperform both* classes of these "clean models", for *all* regimes $\bar{\Theta}$.

---

**Algorithm 1** Dirty Block Sparse

Solve the following convex optimization problem:

$$(\widehat{S}, \widehat{B}) \in \arg\min_{S,B} \quad \frac{1}{2n} \sum_{k=1}^{r} \left\| y^{(k)} - X^{(k)}\left(S^{(k)} + B^{(k)}\right) \right\|_2^2 + \lambda_s \|S\|_{1,1} + \lambda_b \|B\|_{1,\infty}. \tag{1}$$

Then output $\widehat{\Theta} = \widehat{B} + \widehat{S}$.

---

## 3   Main Results and Their Consequences

We now provide precise statements of our main results. A number of recent results have shown that the Lasso [14, 18] and $\ell_1/\ell_\infty$ block-regularization [8] methods succeed in recovering signed supports with controlled error bounds under high-dimensional scaling regimes. Our first two theorems extend these results to our *dirty model* setting. In Theorem 1, we consider the case of deterministic design matrices $X^{(k)}$, and provide sufficient conditions guaranteeing signed support recovery, and elementwise $\ell_\infty$ norm error bounds. In Theorem 2, we specialize this theorem to the case where the

rows of the design matrices are random from a general zero mean Gaussian distribution: this allows us to provide scaling on the number of observations required in order to guarantee signed support recovery and bounded elementwise $\ell_\infty$ norm error.

Our third result is the most interesting in that it explicitly quantifies the performance gains of our method vis-a-vis Lasso and the $\ell_1/\ell_\infty$ block-regularization method. Since this entailed finding the precise constants underlying earlier theorems, and a correspondingly more delicate analysis, we follow Negahban and Wainwright [8] and focus on the case where there are two-tasks (i.e. $r = 2$), and where we have standard Gaussian design matrices as in Theorem 2. Further, while each of two tasks depends on $s$ features, only a fraction $\alpha$ of these are common. It is then interesting to see how the behaviors of the different regularization methods vary with the extent of overlap $\alpha$.

*Comparisons.* Negahban and Wainwright [8] show that there is actually a "phase transition" in the scaling of the probability of successful signed support-recovery with the number of observations. Denote a particular rescaling of the sample-size $\theta_{Lasso}(n, p, \alpha) = \frac{n}{s \log(p-s)}$. Then as Wainwright [18] show, when the rescaled number of samples scales as $\theta_{Lasso} > 2 + \delta$ for any $\delta > 0$, Lasso succeeds in recovering the signed support of all columns with probability converging to one. But when the sample size scales as $\theta_{Lasso} < 2 - \delta$ for any $\delta > 0$, Lasso *fails* with probability converging to one. For the $\ell_1/\ell_\infty$-regularized multiple linear regression, define a similar rescaled sample size $\theta_{1,\infty}(n, p, \alpha) = \frac{n}{s \log(p-(2-\alpha)s)}$. Then as Negahban and Wainwright [8] show there is again a transition in probability of success from near zero to near one, at the rescaled sample size of $\theta_{1,\infty} = (4 - 3\alpha)$. Thus, for $\alpha < 2/3$ ("less sharing") Lasso would perform better since its transition is at a smaller sample size, while for $\alpha > 2/3$ ("more sharing") the $\ell_1/\ell_\infty$ regularized method would perform better.

As we show in our third theorem, the phase transition for our method occurs at the rescaled sample size of $\theta_{1,\infty} = (2 - \alpha)$, which is *strictly* before either the Lasso or the $\ell_1/\ell_\infty$ regularized method except for the boundary cases: $\alpha = 0$, i.e. the case of no sharing, where we *match* Lasso, and for $\alpha = 1$, i.e. full sharing, where we *match* $\ell_1/\ell_\infty$. Everywhere else, we *strictly outperform both* methods. Figure 3 shows the empirical performance of each of the three methods; as can be seen, they agree very well with the theoretical analysis. (Further details in the experiments Section 4).

## 3.1 Sufficient Conditions for Deterministic Designs

We first consider the case where the design matrices $X^{(k)}$ for $k = 1, \cdots, r$ are deterministic, and start by specifying the assumptions we impose on the model. We note that similar sufficient conditions for the deterministic $X^{(k)}$'s case were imposed in papers analyzing Lasso [18] and block-regularization methods [8, 10].

**A0** *Column Normalization* $\left\| X_j^{(k)} \right\|_2 \le \sqrt{2n}$ for all $j = 1, \ldots, p$, $k = 1, \ldots, r$.

Let $\mathcal{U}_k$ denote the support of the $k$-th column of $\bar\Theta$, and $\mathcal{U} = \bigcup_k \mathcal{U}_k$ denote the union of supports for each task. Then we require that

**A1** *Incoherence Condition* $\gamma_b := 1 - \max_{j \in \mathcal{U}^c} \sum_{k=1}^{r} \left\| \left\langle X_j^{(k)}, X_{\mathcal{U}_k}^{(k)} \left( \left\langle X_{\mathcal{U}_k}^{(k)}, X_{\mathcal{U}_k}^{(k)} \right\rangle \right)^{-1} \right\rangle \right\|_1 > 0$.

We will also find it useful to define $\gamma_s := 1 - \max_{1 \le k \le r} \max_{j \in \mathcal{U}_k^c} \left\| \left\langle X_j^{(k)}, X_{\mathcal{U}_k}^{(k)} \right\rangle \left( \left\langle X_{\mathcal{U}_k}^{(k)}, X_{\mathcal{U}_k}^{(k)} \right\rangle \right)^{-1} \right\|_1$. Note that by the incoherence condition **A1**, we have $\gamma_s > 0$.

**A2** *Eigenvalue Condition* $C_{min} := \min_{1 \le k \le r} \lambda_{min} \left( \frac{1}{n} \left\langle X_{\mathcal{U}_k}^{(k)}, X_{\mathcal{U}_k}^{(k)} \right\rangle \right) > 0$.

**A3** *Boundedness Condition* $D_{max} := \max_{1 \le k \le r} \left\| \left( \frac{1}{n} \left\langle X_{\mathcal{U}_k}^{(k)}, X_{\mathcal{U}_k}^{(k)} \right\rangle \right)^{-1} \right\|_{\infty,1} < \infty$.

Further, we require the regularization penalties be set as

$$\lambda_s > \frac{2(2 - \gamma_s)\sigma\sqrt{\log(pr)}}{\gamma_s\sqrt{n}} \qquad \text{and} \qquad \lambda_b > \frac{2(2 - \gamma_b)\sigma\sqrt{\log(pr)}}{\gamma_b\sqrt{n}}. \tag{2}$$

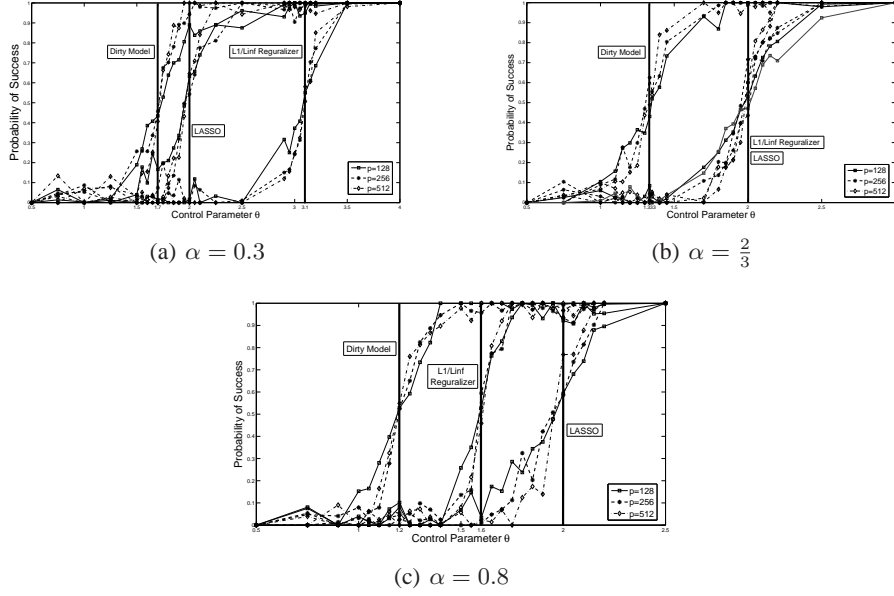

(a) $\alpha = 0.3$

(b) $\alpha = \frac{2}{3}$

(c) $\alpha = 0.8$

Figure 1: Probability of success in recovering the true signed support using dirty model, Lasso and $\ell_1/\ell_\infty$ regularizer. For a 2-task problem, the probability of success for different values of feature-overlap fraction $\alpha$ is plotted. As we can see in the regimes that Lasso is better than, as good as and worse than $\ell_1/\ell_\infty$ regularizer ((a), (b) and (c) respectively), the dirty model outperforms both of the methods, i.e., it requires less number of observations for successful recovery of the true signed support compared to Lasso and $\ell_1/\ell_\infty$ regularizer. Here $s = \lfloor \frac{p}{10} \rfloor$ always.

**Theorem 1.** *Suppose* **A0-A3** *hold, and that we obtain estimate* $\widehat{\Theta}$ *from our algorithm with regularization parameters chosen according to* (2)*. Then, with probability at least* $1 - c_1 \exp(-c_2 n) \to 1$*, we are guaranteed that the convex program* (1) *has a unique optimum and*

*(a) The estimate* $\widehat{\Theta}$ *has no false inclusions, and has bounded* $\ell_\infty$ *norm error so that*

$$Supp(\widehat{\Theta}) \subseteq Supp(\bar{\Theta}), \quad and \quad \|\widehat{\Theta} - \bar{\Theta}\|_{\infty,\infty} \le \underbrace{\sqrt{\frac{4\sigma^2 \log(pr)}{n\, C_{min}}} + \lambda_s D_{max}}_{b_{min}} .$$

*(b)* $sign(Supp(\widehat{\Theta})) = sign\left(Supp(\bar{\Theta})\right)$ *provided that* $\displaystyle\min_{(j,k)\in Supp(\bar{\Theta})} \left|\bar{\theta}_j^{(k)}\right| > b_{min}$.

*Here the positive constants* $c_1, c_2$ *depend only on* $\gamma_s, \gamma_b, \lambda_s, \lambda_b$ *and* $\sigma$*, but are otherwise independent of* $n, p, r$*, the problem dimensions of interest.*

**Remark:** Condition (a) guarantees that the estimate will have no *false inclusions*; i.e. all included features will be relevant. If in addition, we require that it have no *false exclusions* and that recover the support exactly, we need to impose the assumption in (b) that the non-zero elements are large enough to be detectable above the noise.

### 3.2 General Gaussian Designs

Often the design matrices consist of samples from a Gaussian ensemble. Suppose that for each task $k = 1, \ldots, r$ the design matrix $X^{(k)} \in \mathbb{R}^{n \times p}$ is such that each row $X_i^{(k)} \in \mathbb{R}^p$ is a zero-mean Gaussian random vector with covariance matrix $\Sigma^{(k)} \in \mathbb{R}^{p \times p}$, and is independent of every other row. Let $\Sigma_{\mathcal{V},\mathcal{U}}^{(k)} \in \mathbb{R}^{|\mathcal{V}| \times |\mathcal{U}|}$ be the submatrix of $\Sigma^{(k)}$ with rows corresponding to $\mathcal{V}$ and columns to $\mathcal{U}$. We require these covariance matrices to satisfy the following conditions:

**C1** *Incoherence Condition* $\gamma_b := 1 - \displaystyle\max_{j \in \mathcal{U}^c} \sum_{k=1}^{r} \left\| \Sigma_{j,\mathcal{U}_k}^{(k)} \left( \Sigma_{\mathcal{U}_k,\mathcal{U}_k}^{(k)} \right)^{-1} \right\|_1 > 0$

**C2** *Eigenvalue Condition* $C_{min} := \min_{1 \le k \le r} \lambda_{min}\left(\Sigma_{\mathcal{U}_k,\mathcal{U}_k}^{(k)}\right) > 0$ so that the minimum eigenvalue is bounded away from zero.

**C3** *Boundedness Condition* $D_{max} := \left\|\left(\Sigma_{\mathcal{U}_k,\mathcal{U}_k}^{(k)}\right)^{-1}\right\|_{\infty,1} < \infty$.

These conditions are analogues of the conditions for deterministic designs; they are now imposed on the covariance matrix of the (randomly generated) rows of the design matrix.

Further, defining $s := \max_k |\mathcal{U}_k|$, we require the regularization penalties be set as

$$\lambda_s > \frac{\left(4\sigma^2 C_{min}\log(pr)\right)^{1/2}}{\gamma_s\sqrt{nC_{min}} - \sqrt{2s\log(pr)}} \qquad \text{and} \qquad \lambda_b > \frac{\left(4\sigma^2 C_{min}r(r\log(2)+\log(p))\right)^{1/2}}{\gamma_b\sqrt{nC_{min}} - \sqrt{2sr(r\log(2)+\log(p))}}. \qquad (3)$$

**Theorem 2.** *Suppose assumptions* **C1-C3** *hold, and that the number of samples scale as* $n > \max\left(\frac{2s\log(pr)}{C_{min}\gamma_s^2}, \frac{2sr\left(r\log(2)+\log(p)\right)}{C_{min}\gamma_b^2}\right)$. *Suppose we obtain estimate* $\widehat{\Theta}$ *from algorithm* (3). *Then, with probability at least* $1 - c_1\exp\left(-c_2\left(r\log(2)+\log(p)\right)\right) - c_3\exp(-c_4\log(rs)) \to 1$ *for some positive numbers* $c_1 - c_4$, *we are guaranteed that the algorithm estimate* $\widehat{\Theta}$ *is unique and satisfies the following conditions:*

*(a) the estimate* $\widehat{\Theta}$ *has no false inclusions, and has bounded* $\ell_\infty$ *norm error so that*

$$Supp(\widehat{\Theta}) \subseteq Supp(\bar{\Theta}), \quad and \quad \|\widehat{\Theta} - \bar{\Theta}\|_{\infty,\infty} \le \underbrace{\sqrt{\frac{50\sigma^2\log(rs)}{nC_{min}}} + \lambda_s\left(\frac{4s}{C_{min}\sqrt{n}} + D_{max}\right)}_{g_{min}}.$$

*(b)* $sign(Supp(\widehat{\Theta})) = sign\left(Supp(\bar{\Theta})\right)$ *provided that* $\min_{(j,k)\in Supp(\bar{\Theta})}\left|\bar{\theta}_j^{(k)}\right| > g_{min}$.

### 3.3 Sharp Transition for $2$-Task Gaussian Designs

This is one of the most important results of this paper. Here, we perform a more delicate and finer analysis to establish precise quantitative gains of our method. We focus on the special case where $r = 2$ and the design matrix has rows generated from the standard Gaussian distribution $\mathcal{N}(0, I_{n\times n})$, so that $C1 - C3$ hold, with $C_{min} = D_{max} = 1$. As we will see both analytically and experimentally, our method strictly outperforms both Lasso and $\ell_1/\ell_\infty$-block-regularization over for all cases, except at the extreme endpoints of no support sharing (where it matches that of Lasso) and full support sharing (where it matches that of $\ell_1/\ell_\infty$). We now present our analytical results; the empirical comparisons are presented next in Section 4. The results will be in terms of a particular rescaling of the sample size $n$ as

$$\theta(n,p,s,\alpha) := \frac{n}{(2-\alpha)s\log\left(p-(2-\alpha)s\right)}.$$

We will also require the assumptions that

**F1** $\lambda_s > \dfrac{\left(4\sigma^2(1-\sqrt{s/n})(\log(r)+\log(p-(2-\alpha)s))\right)^{1/2}}{(n)^{1/2} - (s)^{1/2} - ((2-\alpha)\ s\ (\log(r)+\log(p-(2-\alpha)s)))^{1/2}}$,

**F2** $\lambda_b > \dfrac{\left(4\sigma^2(1-\sqrt{s/n})r(r\log(2)+\log(p-(2-\alpha)s))\right)^{1/2}}{(n)^{1/2} - (s)^{1/2} - ((1-\alpha/2)\ sr\ (r\log(2)+\log(p-(2-\alpha)s)))^{1/2}}$.

**Theorem 3.** *Consider a 2-task regression problem* $(n,p,s,\alpha)$, *where the design matrix has rows generated from the standard Gaussian distribution* $\mathcal{N}(0, I_{n\times n})$. *Suppose* $\max_{j\in B^*}\left|\left|\Theta_j^{*(1)}\right| -$

$\left|\left|\Theta_j^{*(2)}\right|\right| = o(\lambda_s)$, *where $B^*$ is the submatrix of $\Theta^*$ with rows where both entries are non-zero.*

*Then the estimate $\widehat{\Theta}$ of the problem* (1) *satisfies the following:*

*(**Success**) Suppose the regularization coefficients satisfy $\mathbf{F1} - \mathbf{F2}$. Further, assume that the number of samples scales as $\theta(n, p, s, \alpha) > 1$. Then, with probability at least $1 - c_1 \exp(-c_2 n)$ for some positive numbers $c_1$ and $c_2$, we are guaranteed that $\widehat{\Theta}$ satisfies the support-recovery and $\ell_\infty$ error bound conditions (a-b) in Theorem 2.*

*(**Failure**) If $\theta(n, p, s, \alpha) < 1$ there is no solution $(\hat{B}, \hat{S})$ for any choices of $\lambda_s$ and $\lambda_b$ such that $sign\left(Supp(\widehat{\Theta})\right) = sign\left(Supp(\bar{\Theta})\right)$.*

We note that we require the gap $\left|\left|\left|\Theta_j^{*(1)}\right| - \left|\Theta_j^{*(2)}\right|\right|\right|$ to be small only on rows where both entries are non-zero. As we show in a more general theorem in the appendix, even in the case where the gap is large, the dependence of the sample scaling on the gap is quite weak.

## 4   Empirical Results

In this section, we investigate the performance of our dirty block sparse estimator on synthetic and real-world data. The synthetic experiments explore the accuracy of Theorem 3, and compare our estimator with LASSO and the $\ell_1/\ell_\infty$ regularizer. We see that Theorem 3 is very accurate indeed. Next, we apply our method to a real world datasets containing hand-written digits for classification. Again we compare against LASSO and the $\ell_1/\ell_\infty$.

(a multi-task regression dataset) with $r = 2$ tasks. In both of this real world dataset, we show that dirty model outperforms both LASSO and $\ell_1/\ell_\infty$ practically. For each method, the parameters are chosen via cross-validation; see supplemental material for more details.

### 4.1   Synthetic Data Simulation

We consider a $r = 2$-task regression problem as discussed in Theorem 3, for a range of parameters $(n, p, s, \alpha)$. The design matrices $X$ have each entry being i.i.d. Gaussian with mean 0 and variance 1. For each fixed set of $(n, s, p, \alpha)$, we generate 100 instances of the problem. In each instance, given $p, s, \alpha$, the locations of the non-zero entries of the true $\bar{\Theta}$ are chosen at randomly; each non-zero entry is then chosen to be i.i.d. Gaussian with mean 0 and variance 1. $n$ samples are then generated from this. We then attempt to estimate using three methods: our dirty model, $\ell_1/\ell_\infty$ regularizer and LASSO. In each case, and for each instance, the penalty regularizer coefficients are found by cross validation. After solving the three problems, we compare the signed support of the solution with the true signed support and decide whether or not the program was successful in signed support recovery. We describe these process in more details in this section.

**Performance Analysis**: We ran the algorithm for five different values of the overlap ratio $\alpha \in \{0.3, \frac{2}{3}, 0.8\}$ with three different number of features $p \in \{128, 256, 512\}$. For any instance of the problem $(n, p, s, \alpha)$, if the recovered matrix $\hat{\Theta}$ has the same sign support as the true $\bar{\Theta}$, then we count it as success, otherwise failure (even if one element has different sign, we count it as failure).

As Theorem 3 predicts and Fig 3 shows, the right scaling for the number of oservations is $\frac{n}{s \log(p - (2-\alpha)s)}$, where all curves stack on the top of each other at $2 - \alpha$. Also, the number of observations required by dirty model for true signed support recovery is always less than both LASSO and $\ell_1/\ell_\infty$ regularizer. Fig 1(a) shows the probability of success for the case $\alpha = 0.3$ (when LASSO is better than $\ell_1/\ell_\infty$ regularizer) and that dirty model outperforms both methods. When $\alpha = \frac{2}{3}$ (see Fig 1(b)), LASSO and $\ell_1/\ell_\infty$ regularizer performs the same; but dirty model require almost 33% less observations for the same performance. As $\alpha$ grows toward 1, e.g. $\alpha = 0.8$ as shown in Fig 1(c), $\ell_1/\ell_\infty$ performs better than LASSO. Still, dirty model performs better than both methods in this case as well.

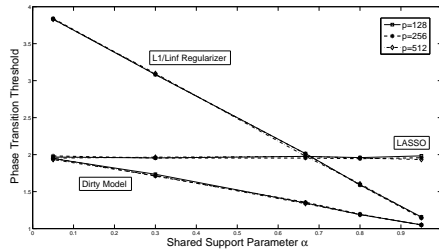

Figure 2: Verification of the result of the Theorem 3 on the behavior of phase transition threshold by changing the parameter $\alpha$ in a 2-task $(n, p, s, \alpha)$ problem for dirty model, LASSO and $\ell_1/\ell_\infty$ regularizer. The $y$-axis is $\frac{n}{s \log(p - (2 - \alpha)s)}$, where $n$ is the number of samples at which threshold was observed. Here $s = \lfloor \frac{p}{10} \rfloor$. Our dirty model method shows a gain in sample complexity over the entire range of sharing $\alpha$. The pre-constant in Theorem 3 is also validated.

| n | | | Our Model | $\ell_1/\ell_\infty$ | LASSO |
|---|---|---|---|---|---|
| 10 | Average Classification Error | | 8.6% | 9.9% | 10.8% |
| | Variance of Error | | 0.53% | 0.64% | 0.51% |
| | Average Row Support Size | $B$:165 | $B + S$:171 | 170 | 123 |
| | Average Support Size | $S$:18 | $B + S$:1651 | 1700 | 539 |
| 20 | Average Classification Error | | 3.0% | 3.5% | 4.1% |
| | Variance of Error | | 0.56% | 0.62% | 0.68% |
| | Average Row Support Size | $B$:211 | $B + S$:226 | 217 | 173 |
| | Average Support Size | $S$:34 | $B + S$:2118 | 2165 | 821 |
| 40 | Average Classification Error | | 2.2% | 3.2% | 2.8% |
| | Variance of Error | | 0.57% | 0.68% | 0.85% |
| | Average Row Support Size | $B$:270 | $B + S$:299 | 368 | 354 |
| | Average Support Size | $S$:67 | $B + S$:2761 | 3669 | 2053 |

Table 1: Handwriting Classification Results for our model, $\ell_1/\ell_\infty$ and LASSO

**Scaling Verification**: To verify that the phase transition threshold changes linearly with $\alpha$ as predicted by Theorem 3, we plot the phase transition threshold versus $\alpha$. For five different values of $\alpha \in \{0.05, 0.3, \frac{2}{3}, 0.8, 0.95\}$ and three different values of $p \in \{128, 256, 512\}$, we find the phase transition threshold for dirty model, LASSO and $\ell_1/\ell_\infty$ regularizer. We consider the point where the probability of success in recovery of signed support exceeds $50\%$ as the phase transition threshold. We find this point by interpolation on the closest two points. Fig 2 shows that phase transition threshold for dirty model is always lower than the phase transition for LASSO and $\ell_1/\ell_\infty$ regularizer.

## 4.2 Handwritten Digits Dataset

We use the handwritten digit dataset [1], containing features of handwritten numerals (0-9) extracted from a collection of Dutch utility maps. This dataset has been used by a number of papers [17, 6] as a reliable dataset for handwritten recognition algorithms. There are thus $r = 10$ tasks, and each handwritten sample consists of $p = 649$ features.

Table 1 shows the results of our analysis for different sizes $n$ of the training set . We measure the classification error for each digit to get the 10-vector of errors. Then, we find the average error and the variance of the error vector to show how the error is distributed over all tasks. We compare our method with $\ell_1/\ell_\infty$ regularizer method and LASSO. Again, in all methods, parameters are chosen via cross-validation.

For our method we separate out the $B$ and $S$ matrices that our method finds, so as to illustrate how many features it identifies as "shared" and how many as "non-shared". For the other methods we just report the straight row and support numbers, since they do not make such a separation.

## Acknowledgements

We acknowledge support from NSF grant IIS-101842, and NSF CAREER program, Grant 0954059.

# References

[1] A. Asuncion and D.J. Newman. UCI Machine Learning Repository, http://www.ics.uci.edu/ mlearn/MLRepository.html. University of California, School of Information and Computer Science, Irvine, CA, 2007.

[2] F. Bach. Consistency of the group lasso and multiple kernel learning. *Journal of Machine Learning Research*, 9:1179–1225, 2008.

[3] R. Baraniuk. Compressive sensing. *IEEE Signal Processing Magazine*, 24(4):118–121, 2007.

[4] R. Caruana. Multitask learning. *Machine Learning*, 28:41–75, 1997.

[5] C.Zhang and J.Huang. Model selection consistency of the lasso selection in high-dimensional linear regression. *Annals of Statistics*, 36:1567–1594, 2008.

[6] X. He and P. Niyogi. Locality preserving projections. In *NIPS*, 2003.

[7] K. Lounici, A. B. Tsybakov, M. Pontil, and S. A. van de Geer. Taking advantage of sparsity in multi-task learning. In *22nd Conference On Learning Theory (COLT)*, 2009.

[8] S. Negahban and M. J. Wainwright. Joint support recovery under high-dimensional scaling: Benefits and perils of $\ell_{1,\infty}$-regularization. In *Advances in Neural Information Processing Systems (NIPS)*, 2008.

[9] S. Negahban and M. J. Wainwright. Estimation of (near) low-rank matrices with noise and high-dimensional scaling. In *ICML*, 2010.

[10] G. Obozinski, M. J. Wainwright, and M. I. Jordan. Support union recovery in high-dimensional multivariate regression. *Annals of Statistics*, 2010.

[11] P. Ravikumar, H. Liu, J. Lafferty, and L. Wasserman. Sparse additive models. *Journal of the Royal Statistical Society, Series B*.

[12] P. Ravikumar, M. J. Wainwright, and J. Lafferty. High-dimensional ising model selection using $\ell_1$-regularized logistic regression. *Annals of Statistics*, 2009.

[13] B. Recht, M. Fazel, and P. A. Parrilo. Guaranteed minimum-rank solutions of linear matrix equations via nuclear norm minimization. In *Allerton Conference, Allerton House, Illinois*, 2007.

[14] R. Tibshirani. Regression shrinkage and selection via the lasso. *Journal of the Royal Statistical Society, Series B*, 58(1):267–288, 1996.

[15] J. A. Tropp, A. C. Gilbert, and M. J. Strauss. Algorithms for simultaneous sparse approximation. *Signal Processing, Special issue on "Sparse approximations in signal and image processing"*, 86:572–602, 2006.

[16] B. Turlach, W.N. Venables, and S.J. Wright. Simultaneous variable selection. *Techno- metrics*, 27:349–363, 2005.

[17] M. van Breukelen, R.P.W. Duin, D.M.J. Tax, and J.E. den Hartog. Handwritten digit recognition by combined classifiers. *Kybernetika*, 34(4):381–386, 1998.

[18] M. J. Wainwright. Sharp thresholds for noisy and high-dimensional recovery of sparsity using $\ell_1$-constrained quadratic programming (lasso). *IEEE Transactions on Information Theory*, 55: 2183–2202, 2009.

